# Empirical Bernstein Inequalities for U-Statistics

**Thomas Peel**
LIF, Aix-Marseille Université
39, rue F. Joliot Curie
F-13013 Marseille, France
thomas.peel@lif.univ-mrs.fr

**Sandrine Anthoine**
LATP, Aix-Marseille Université, CNRS
39, rue F. Joliot Curie
F-13013 Marseille, France
anthoine@cmi.univ-mrs.fr

**Liva Ralaivola**
LIF, Aix-Marseille Université
39, rue F. Joliot Curie
F-13013 Marseille, France
liva.ralaivola@lif.univ-mrs.fr

## Abstract

We present original empirical Bernstein inequalities for U-statistics with bounded symmetric kernels $q$. They are expressed with respect to empirical estimates of either the variance of $q$ or the conditional variance that appears in the Bernstein-type inequality for U-statistics derived by Arcones [2]. Our result subsumes other existing empirical Bernstein inequalities, as it reduces to them when U-statistics of order 1 are considered. In addition, it is based on a rather direct argument using two applications of the same (non-empirical) Bernstein inequality for U-statistics. We discuss potential applications of our new inequalities, especially in the realm of learning ranking/scoring functions. In the process, we exhibit an efficient procedure to compute the variance estimates for the special case of bipartite ranking that rests on a sorting argument. We also argue that our results may provide test set bounds and particularly interesting empirical racing algorithms for the problem of online learning of scoring functions.

## 1 Introduction

The motivation of the present work lies in the growing interest of the machine learning community for learning tasks that are richer than now well-studied classification and regression. Among those, we especially have in mind the task of *ranking*, where one is interested in learning a ranking function capable of predicting an accurate ordering of objects according to some attached relevance information. Tackling such problems generally implies the use of loss functions other than the 0-1 misclassification loss such as, for example, a *misranking loss* [6] or a surrogate thereof. For $(\mathbf{x}, y)$ and $(\mathbf{x}', y')$ two pairs from some space $\mathcal{Z} := \mathcal{X} \times \mathcal{Y}$ (e.g., $\mathcal{X} = \mathbb{R}^d$ and $\mathcal{Y} = \mathbb{R}$) the misranking loss $\ell^{\mathrm{rank}}$ and a surrogate convex loss $\ell^{\mathrm{sur}}$ may be defined for a scoring function $f \in \mathcal{Y}^{\mathcal{X}}$ as:

$$\ell^{\mathrm{rank}}(f, (\mathbf{x}, y), (\mathbf{x}', y')) := \mathbb{1}_{\{(y-y')(f(\mathbf{x})-f(\mathbf{x}'))<0\}}, \tag{1}$$

$$\ell^{\mathrm{sur}}(f, (\mathbf{x}, y), (\mathbf{x}', y')) := (1 - (y - y')(f(\mathbf{x}) - f(\mathbf{x}')))^2. \tag{2}$$

Given such losses or, more generally, a loss $\ell : \mathcal{Y}^{\mathcal{X}} \times \mathcal{Z} \times \mathcal{Z} \to \mathbb{R}$, and a training sample $\underline{Z}_n = \{(X_i, Y_i)\}_{i=1}^n$ of *independent* copies of some random variable $Z := (X, Y)$ distributed according to $D$, the learning task is to derive a function $f \in \mathcal{X}^{\mathcal{Y}}$ such that the *expected risk* $R_\ell(f)$ of $f$

$$R_\ell(f) := \mathbb{E}_{Z, Z' \sim D} \ell(f, Z, Z') = \mathbb{E}_{Z, Z' \sim D} \ell(f, (X, Y), (X', Y'))$$

is as small as possible. In practice, this naturally brings up the empirical estimate $\hat{R}_\ell(f, \underline{Z}_n)$

$$\hat{R}_\ell(f, \underline{Z}_n) := \frac{1}{n(n-1)} \sum_{i \neq j} \ell(f, (X_i, Y_i), (X_j, Y_j)), \tag{3}$$

which is a *U-statistic* [6, 10].

An important question is to precisely characterize how $\hat{R}_\ell(f, \underline{Z}_n)$ is related to $R_\ell(f)$ and, more specifically, one may want to derive an upper bound on $R_\ell(f)$ that is expressed in terms of $\hat{R}_\ell(f, \underline{Z}_n)$ and other quantities such as a measure of the capacity of the class of functions $f$ belongs to and the size $n$ of $\underline{Z}_n$ – in other words, we may talk about generalization bounds [4]. Pivotal tools to perform such analysis are *tail/concentration inequalities*, which say how probable it is for a function of several independent variables to deviate from its expectation; of course, the sharper the concentration inequalities the more accurate the characterization of the relation between the empirical estimate and its expectation. It is therefore of the utmost importance to have at hand tail inequalities that are sharp; it is just as important that these inequalities rely as much as possible on empirical quantities.

Here, we propose new *empirical Bernstein inequalities* for U-statistics. As indicated by the name (i) our results are Bernstein-type inequalities and therefore make use of information on the variance of the variables under consideration, (ii) instead of resting on some assumed knowledge about this variance, they only rely on empirical related quantities and (iii) they apply to U-statistics. Our new inequalities generalize those of [3] and [13], which also feature points (i) and (ii) (but *not* (iii)), while based on simple arguments. To the best of our knowledge, these are the first results that fulfill (i), (ii) and (iii); they may give rise to a few applications, of which we describe two in the sequel.

The paper is organized as follows. Section 2 introduces the notations and briefly recalls the basics of U-statistics as well as tail inequalities our results are based upon. Our empirical Bernstein inequalities are presented in Section 3; we also provide an efficient way of computing the empirical variance when the U-statistics considered are based on the misranking loss $\ell^{\text{rank}}$ of (1). Section 4 discusses two applications of our new results: test set bounds for bipartite ranking and online ranking.

## 2 Background

### 2.1 Notation

The following notation will hold from here on. $Z$ is a random variable of distribution $D$ taking values in $\mathcal{Z} := \mathcal{X} \times \mathcal{Y}$; $Z', Z_1, \ldots, Z_n$ are independent copies of $Z$ and $\underline{Z}_n := \{Z_i = (X_i, Y_i)\}_{i=1}^n$ and $\underline{Z}_{p:q} := \{Z_i\}_{i=p}^q$.

$\mathcal{A}_n^m$ denotes the set $\mathcal{A}_n^m := \{(i_1, \ldots, i_m) : 1 \leq i_1 \neq \ldots \neq i_m \leq n\}$, with $0 \leq m \leq n$.

Finally, a function $q : \mathcal{Z}^m \to \mathbb{R}$ is said to be symmetric if the value of $q(\mathbf{z}) = q(z_1, \ldots, z_m)$ is independent of the order of the $z_i$'s in $\mathbf{z}$.

### 2.2 U-statistics and Tail Inequalities

**Definition 1** (U-statistic, Hoeffding [10])**.** The random variable $\hat{U}_q(\underline{Z}_n)$ defined as

$$\hat{U}_q(\underline{Z}_n) := \frac{1}{|\mathcal{A}_n^m|} \sum_{\mathbf{i} \in \mathcal{A}_n^m} q(Z_{i_1}, \ldots, Z_{i_m}),$$

is a U-statistic of order $m$ with *kernel* $q$, when $q : \mathcal{Z}^m \to \mathbb{R}$ is a measurable function on $\mathcal{Z}^m$.

*Remark* 1. Obviously, $\mathbb{E}_{\underline{Z}_m} q(Z_1, \ldots, Z_m) = \mathbb{E}_{\underline{Z}_n} \hat{U}_q(\underline{Z}_n)$; in addition, $\mathbb{E}_{\underline{Z}_n} \hat{U}_q(\underline{Z}_n)$ is a lowest variance estimate of $\mathbb{E}_{\underline{Z}_m} q(Z_1, \ldots, Z_m)$ based on $\underline{Z}_n$ [10]. Also, reusing some notation from the introduction, $\hat{R}_\ell(f, \underline{Z}_n)$ of Eq. (3) is a U-statistic of order 2 with kernel $q_f(Z, Z') := \ell(f, Z, Z')$.

*Remark* 2. Two peculiarities of U-statistics that entail a special care are the following: (i) they are sums of identically distributed but *dependent* variables: special tools need be resorted to in order to deal with these dependencies to characterize the deviation of $\hat{U}_q(\underline{Z}_n)$ from $\mathbb{E}q$, and (ii) from an algorithmic point of view, their direct computations may be expensive, as it scales as $O(n^m)$; in Section 3, we show for the special case of bipartite ranking how this complexity can be reduced.

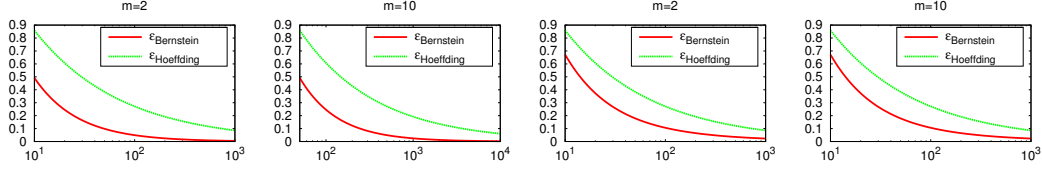

Figure 1: First two plots: values of the right-hand size of (5) and (6), for $D_{\text{uni}}$ and kernel $q_m$ for $m = 2$ and $m = 10$ (see Example 1) as functions of $n$. Last two plots: same for $D_{\text{Ber}}(0.15)$.

We now recall three tail inequalities (Eq. (5), (6), (7)) that hold for U-statistics with *symmetric* and *bounded* kernels $q$. Normally, these inequalities make explicit use of the length $q_{\max} - q_{\min}$ of the range $[q_{\min}, q_{\max}]$ of $q$. To simplify the reading, we will consider without loss of generality that $q$ has range $[0, 1]$ (an easy way of retrieving the results for bounded $q$ is to consider $q/\|q\|_\infty$).

One key quantity that appears in the original versions of tail inequalities (5) and (6) below is $\lfloor n/m \rfloor$, the integer part of the ratio $n/m$ – this quantity might be thought of as the *effective* number of data. To simplify the notation, we will assume that $n$ is a multiple of $m$ and, therefore, $\lfloor n/m \rfloor = (n/m)$.

**Theorem 1** (First order tail inequality for $\hat{U}_q$, [11].). *Hoeffding proved the following:*

$$\forall \varepsilon > 0, \ \mathbb{P}_{\underline{Z}_n} \left\{ \left| \mathbb{E}_{\underline{Z}'_n} \hat{U}_q(\underline{Z}'_n) - \hat{U}_q(\underline{Z}_n) \right| \geq \varepsilon \right\} \leq 2 \exp \left\{ -(n/m)\varepsilon^2 \right\}, \tag{4}$$

*Hence $\forall \delta \in (0, 1]$, with probability at least $1 - \delta$ over the random draw of $\underline{Z}_n$:*

$$\left| \mathbb{E}_{\underline{Z}'_n} \hat{U}_q(\underline{Z}'_n) - \hat{U}_q(\underline{Z}_n) \right| \leq \sqrt{\frac{1}{(n/m)} \ln \frac{2}{\delta}}. \tag{5}$$

To go from the tail inequality (4) to the bound version (5), it suffices to make use of the elementary *inequality reversal lemma* (Lemma 1) provided in section 3, used also for the bounds given below.

**Theorem 2** (Bernstein Inequalities for $\hat{U}_q$, [2, 11]). *Hoeffding [11] and, later, Arcones [2] refined the previous result in the form of Bernstein-type inequalities of the form*

$$\forall \varepsilon > 0, \ \mathbb{P}_{\underline{Z}_n} \left\{ \left| \mathbb{E}_{\underline{Z}'_n} \hat{U}_q(\underline{Z}'_n) - \hat{U}_q(\underline{Z}_n) \right| \geq \varepsilon \right\} \leq a \exp \left\{ -\frac{(n/m)\varepsilon^2}{2\vartheta_{q,m} + b_m \varepsilon} \right\},$$

*For Hoeffding, $a = 2$, $\vartheta_{q,m} = \Sigma_q^2$ where, $\Sigma_q^2$ is the variance of $q(Z_1, \ldots, Z_m)$ and $b_m = 2/3$. Hence, $\forall \delta \in (0, 1]$, with probability at least $1 - \delta$:*

$$\left| \mathbb{E}_{\underline{Z}'_n} \hat{U}_q(\underline{Z}'_n) - \hat{U}_q(\underline{Z}_n) \right| \leq \sqrt{\frac{2\Sigma_q^2}{(n/m)} \ln \frac{2}{\delta}} + \frac{2}{3(n/m)} \ln \frac{2}{\delta}. \tag{6}$$

*For Arcones, $a = 4$, $\vartheta_{q,m} = m\sigma_q^2$ where $\sigma_q^2$ is the variance of $\mathbb{E}_{Z_2,\ldots,Z_m} q(Z_1, Z_2, \ldots, Z_m)$ (this is a function of $Z_1$) and $b_m = 2^{m+3} m^{m-1} + (2/3)m^{-2}$. $\forall \delta \in (0, 1]$, with probability at least $1 - \delta$:*

$$\left| \mathbb{E}_{\underline{Z}'_n} \hat{U}_q(\underline{Z}'_n) - \hat{U}_q(\underline{Z}_n) \right| \leq \sqrt{\frac{2m\sigma_q^2}{(n/m)} \ln \frac{4}{\delta}} + \frac{b_m}{(n/m)} \ln \frac{4}{\delta}. \tag{7}$$

With a slight abuse, we will now refer to Eq. (5), (6) and (7) as tail inequalities. In essence, these are confidence intervals at level $1 - \delta$ for $\mathbb{E}_{\underline{Z}_m} q(\underline{Z}_m) = \mathbb{E}_{\underline{Z}_n} \hat{U}_q(\underline{Z}_n)$.

*Remark* 3. Eq. (7) is based on the so-called Hoeffding decomposition of U-statistics [11]. It provides a more accurate Bernstein-type inequality than that of Eq. (6), as $m\sigma_q^2$ is known to be smaller than $\Sigma_q^2$ (see [16]). However, for moderate values of $n/m$ (e.g. $n/m < 10^5$) and reasonable values of $\delta$ (e.g. $\delta = 0.05$), the influence of the log terms might be such that the advantage of (7) over (6) goes unnoticed. Thus, we detail our results focusing on an empirical version of (6).

*Example* 1. To illustrate how the use of the variance information provides smaller confidence intervals, consider the kernel $q_m := \prod_{i=1}^m z_i$ and two distributions $D_{\text{uni}}$ and $D_{\text{Ber}}(p)$. $D_{\text{uni}}$ is the uniform distribution on $[0, 1]$, for which $\Sigma^2 = \frac{1}{3^m} - \frac{1}{4^m}$. $D_{\text{Ber}}(p)$ is the Bernoulli distribution with parameter $p \in [0, 1]$, for which $\Sigma^2 = p^m(1 - p^m)$. Figure 1 shows the behaviors of (6) and (5) for various values of $m$ as functions of $n$. Observe that the variance information renders the bound smaller.

# 3 Main Results

This section presents the main results of the paper. We first introduce the inequality reversal lemma, which allows to transform tail inequalities into upper bounds (or confidence intervals), as in (5)-(7).

**Lemma 1** (Inequality Reversal lemma). *Let $X$ be a random variable and $a, b > 0, c, d \geq 0$ such that*

$$\forall \varepsilon > 0, \ \mathbb{P}_X(|X| \geq \varepsilon) \leq a \exp\left\{-\frac{b\varepsilon^2}{c + d\varepsilon}\right\}, \tag{8}$$

*then, with probability at least $1 - \delta$*

$$|X| \leq \sqrt{\frac{c}{b} \ln \frac{a}{\delta}} + \frac{d}{b} \ln \frac{a}{\delta}. \tag{9}$$

*Proof.* Solving for $\varepsilon$ such that the right hand side of (8) is equal to $\delta$ gives:

$$\varepsilon = \frac{1}{2b}\left(d \ln \frac{a}{\delta} + \sqrt{d^2 \ln^2 \frac{a}{\delta} + 4bc \ln \frac{a}{\delta}}\right).$$

Using $\sqrt{a + b} \leq \sqrt{a} + \sqrt{b}$ gives an upper bound on $\varepsilon$ and provides the result. $\square$

## 3.1 Empirical Bernstein Inequalities

Let us now define the empirical variances we will use in our main result.

**Definition 2.** Let $\hat{\Sigma}_q^2$ be the U-statistic of order $2m$ defined as:

$$\hat{\Sigma}_q^2(\underline{Z}_n) := \frac{1}{|\mathcal{A}_n^{2m}|} \sum_{\mathbf{i} \in \mathcal{A}_n^{2m}} \left(q(Z_{i_1}, \ldots, Z_{i_m}) - q(Z_{i_{m+1}}, \ldots, Z_{i_{2m}})\right)^2, \tag{10}$$

and $\hat{\sigma}_q^2$ be the U-statistic of order $2m - 1$ defined as:

$$\hat{\sigma}_q^2(\underline{Z}_n) := \frac{1}{|\mathcal{A}_n^{2m-1}|} \sum_{\mathbf{i} \in \mathcal{A}_n^{2m-1}} q(Z_{i_1}, Z_{i_2}, \ldots, Z_{i_m}) q(Z_{i_1}, Z_{i_{m+1}}, \ldots, Z_{i_{2m-1}}), \tag{11}$$

It is straightforward to see that (cf. the definitions of $\Sigma_q^2$ in (6) and $\sigma_q^2$ in (7))

$$\mathbb{E}_{\underline{Z}_n} \hat{\Sigma}_q^2(\underline{Z}_n) = \Sigma_q^2, \quad \text{and} \quad \mathbb{E}_{\underline{Z}_n} \hat{\sigma}_q^2(\underline{Z}_n) = \sigma_q^2 + \mathbb{E}_{\underline{Z}_m}^2 q(Z_1, \ldots, Z_m).$$

We have the following main result.

**Theorem 3** (Empirical Bernstein Inequalities/Bounds). *With probability at least $1 - \delta$ over $\underline{Z}_n$,*

$$\left|\mathbb{E}_{\underline{Z}_n'} \hat{U}_q(\underline{Z}_n') - \hat{U}_q(\underline{Z}_n)\right| \leq \sqrt{\frac{2\hat{\Sigma}_q^2}{(n/m)} \ln \frac{4}{\delta}} + \frac{5}{(n/m)} \ln \frac{4}{\delta}. \tag{12}$$

*And, also, with probability at least $1 - \delta$, ($b_m$ is the same as in (7))*

$$\left|\mathbb{E}_{\underline{Z}_n'} \hat{U}_q(\underline{Z}_n') - \hat{U}_q(\underline{Z}_n)\right| \leq \sqrt{\frac{2m\hat{\sigma}_q^2}{(n/m)} \ln \frac{8}{\delta}} + \frac{5\sqrt{m} + b_m}{(n/m)} \ln \frac{8}{\delta}. \tag{13}$$

*Proof.* We provide the proof of (12) for the upper bound of the confidence interval; the same reasoning carries over to prove the lower bound. The proof of (13) is very similar.

First, let us call $Q$ the kernel of $\hat{\Sigma}_q^2$:

$$Q(Z_1, \ldots, Z_{2m}) := \left(q(Z_1, \ldots, Z_m) - q(Z_{m+1}, \ldots, Z_{2m})\right)^2.$$

$Q$ is of order $2m$, has range $[0,1]$ but it is not necessarily symmetric. An equivalent symmetric kernel for $\hat{\Sigma}_q^2$ is $Q_{\mathrm{sym}}$:

$$Q_{\mathrm{sym}}(Z_1,\ldots,Z_{2m}) := \frac{1}{(2m)!}\sum_{\omega\in\mathcal{P}_{2m}}\big(q(Z_{\omega(1)},\ldots,Z_{\omega(m)}) - q(Z_{\omega(m+1)},\ldots,Z_{\omega(2m)})\big)^2$$

where $\mathcal{P}_m$ is the set of all the permutations over $\{1,\ldots,m\}$. This kernel is symmetric (and has range $[0,1]$) and Theorem 2 can be applied to bound $\Sigma^2$ as follows: with prob. at least $1-\delta$

$$\Sigma^2 = \mathbb{E}_{\underline{Z}'_{2m}}Q_{\mathrm{sym}}(\underline{Z}'_{2m}) = \mathbb{E}_{\underline{Z}'_n}\hat{\Sigma}_q^2(\underline{Z}'_n) \leq \hat{\Sigma}_q^2(\underline{Z}_n) + \sqrt{\frac{2\mathbb{V}(Q_{\mathrm{sym}})}{(n/2m)}\ln\frac{2}{\delta}} + \frac{2}{3(n/2m)}\ln\frac{2}{\delta},$$

where $\mathbb{V}(Q_{\mathrm{sym}})$ is the variance of $Q_{\mathrm{sym}}$. As $Q_{\mathrm{sym}}$ has range $[0,1]$,

$$\mathbb{V}(Q_{\mathrm{sym}}) = \mathbb{E}Q_{\mathrm{sym}}^2 - \mathbb{E}^2 Q_{\mathrm{sym}} \leq \mathbb{E}Q_{\mathrm{sym}}^2 \leq \mathbb{E}Q_{\mathrm{sym}} = \Sigma^2,$$

and therefore

$$\Sigma^2 \leq \hat{\Sigma}_q^2(\underline{Z}_n) + \sqrt{\frac{4\Sigma^2}{(n/m)}\ln\frac{2}{\delta}} + \frac{4}{3(n/m)}\ln\frac{2}{\delta}.$$

(To establish (13) we additionally use $\hat{\sigma}_q^2(\underline{Z}_n) \geq \sigma_q^2$).

Following the approach of [13], we introduce $\left(\sqrt{\Sigma^2} - \sqrt{(m/n)\ln(2/\delta)}\right)^2$ and we get

$$\left(\sqrt{\Sigma^2} - \sqrt{(m/n)\ln(2/\delta)}\right)^2 \leq \hat{\Sigma}_q^2(\underline{Z}_n) + \frac{7}{3(n/m)}\ln\frac{2}{\delta},$$

and taking the square root of both side, using $1+\sqrt{7/3} < 3$ and $\sqrt{a+b} \leq \sqrt{a}+\sqrt{b}$ again gives

$$\sqrt{\Sigma^2} \leq \sqrt{\hat{\Sigma}_q^2(\underline{Z}_n)} + 3\sqrt{\frac{1}{(n/m)}\ln\frac{2}{\delta}}.$$

We now apply Theorem 2 to bound $|\mathbb{E}_{\underline{Z}'_n}\hat{U}_q(\underline{Z}'_n) - \hat{U}_q(\underline{Z}_n)|$, and plug in the latter equation, adjusting $\delta$ to $\delta/2$ so the obtained inequality still holds with probability $1-\delta$. Bounding appropriate constants gives the desired result. $\qquad\square$

*Remark* 4. In addition to providing an empirical Bernstein bound for U-statistics based on arbitrary bounded kernels, our result differs from that of Maurer and Pontil [13] by the way we derive it. Here, we apply the same tail inequality twice, taking advantage of the fact that estimates for the variances we are interested in are also U-statistics. Maurer and Pontil use a tail inequality on self bounded random variables and do not explicitly take advantage of the estimates they use being U-statistics.

### 3.2 Efficient Computation of the Variance Estimate for Bipartite Ranking

We have just showed how empirical Bernstein inequalities can be derived for U-statistics. The estimates that enter into play in the presented results are U-statistics with kernels of order $2m$ (or $2m-1$), meaning that a direct approach to practically compute them would scale as $O(n^{2m})$ (or $O(n^{2m-1})$). This scaling might be prohibitive as soon as $n$ gets large.

Here, we propose an efficient way of evaluating the estimate $\hat{\Sigma}_q^2$ (a similar reasoning carries over for $\hat{\sigma}_q^2$) in the special case where $\mathcal{Y} = \{-1,+1\}$ and the kernel $q_f$ induces the misranking loss (1):

$$q_f((x,y),(x',y')) := \mathbb{1}_{\{(y-y')(f(x)-f(x'))<0\}}, \ \forall f\in\mathbb{R}^{\mathcal{X}},$$

which is a symmetric kernel of order $m=2$ with range $[0,1]$. In other words, we address the *bipartite ranking* problem. We have the following result.

**Proposition 1** (Efficient computation of $\hat{\Sigma}_{q_f}^2$). $\forall n$, *the computation of*

$$\hat{\Sigma}_{q_f}(\mathbf{z}_n) = \frac{1}{|\mathcal{A}_n^4|}\sum_{\mathbf{i}\in\mathcal{A}_n^4}\left(\mathbb{1}_{\left\{(y_{i_1}-y_{i_2})(f(x_{i_1})-f(x_{i_2}))<0\right\}} - \mathbb{1}_{\left\{(y_{i_3}-y_{i_4})(f(x_{i_3})-f(x_{i_4}))<0\right\}}\right)^2$$

*can be performed in $O(n\ln n)$.*

*Proof.* We simply provide an algorithmic way to compute $\hat{\Sigma}_{q_f}^2(\mathbf{z}_n)$. To simplify the reading, we replace $i_1, i_2, i_3, i_4$ by $i, j, k, l$, respectively. We also drop the normalization factor $|\mathcal{A}_n^4|^{-1}$ (hence the use of $\propto$ instead of $=$ in the first line below). We have

$$\hat{\Sigma}_{q_f}^2(\mathbf{z}_n) \propto \sum_{\substack{i,j,k,l \\ i \neq j \neq k \neq l}} (q_f(z_i, z_j) - q_f(z_k, z_l))^2 = \sum_{\substack{i,j,k,l \\ i \neq j \neq k \neq l}} \left( q_f^2(z_i, z_j) - 2q_f(z_i, z_j)q_f(z_k, z_l) + q_f^2(z_k, z_l) \right),$$

$$= 2(n-2)(n-3) \sum_{i,j} q_f(z_i, z_j) - 2 \sum_{\substack{i,j,k,l \\ i \neq j \neq k \neq l}} q_f(z_i, z_j)q_f(z_k, z_l) \quad \text{since} \left( \begin{smallmatrix} q_f^2 = q_f \\ q_f(\mathbf{z}, \mathbf{z}) = 0 \end{smallmatrix} \right).$$

The first term of the last line is proportional to the well-known Wilcoxon-Mann-Whitney statistic [9]. There exist efficient ways ($O(n \ln n)$) to compute it, based on sorting the values of the $f(x_i)$'s. We show how to deal with the second term, using sorting arguments as well. Note that

$$\sum_{\substack{i,j,k,l \\ i \neq j \neq k \neq l}} q_f(z_i, z_j)q_f(z_k, z_l) = \left( \sum_{i,j} q_f(z_i, z_j) \right)^2 - 4 \sum_{i \neq j \neq k} q_f(z_i, z_j)q_f(z_i, z_k) - 2 \sum_{i,j} q_f^2(z_i, z_j).$$

We have subtracted from the square of $\sum_{i,j} q_f(z_i, z_j)$ all the products $q_f(z_i, z_j)q_f(z_k, z_l)$ such that exactly one of the variables appears both in $q_f(z_i, z_j)$ and $q_f(z_k, z_l)$, which happens when $i = k$, $i = l, j = k, j = l$; using the symmetry of $q_f$ then provides the second term (together with the factor 4). We also have subtracted all the products $q_f(z_i, z_j)q_f(z_k, z_l)$ where $i = k$ and $j = l$ or $i = l$ and $j = k$, in which case the product reduces to $q_f^2(z_i, z_j)$ (hence the factor 2) – this gives the last term. Thus (using $q_f^2 = q_f$), defining $R(\mathbf{z}_n) := \sum_{ij} q_f(z_i, z_j)$ and doing some simple calculations:

$$\hat{\Sigma}_{q_f}(\mathbf{z}_n) = \frac{1}{|\mathcal{A}_n^4|} \left[ -2R^2(\mathbf{z}_n) + 2(n^2 - 5n + 8)R(\mathbf{z}_n) + 8 \sum_{i \neq j \neq k} q_f(z_i, z_j)q_f(z_i, z_k) \right] \qquad (14)$$

The only term that now requires special care is the last one (which is proportional to $\hat{\sigma}_{q_f}^2(\mathbf{z}_n)$).

Recalling that $q_f(z_i, z_j) = \mathbb{1}_{\{(y_i - y_j)(f(x_i) - f(x_j)) < 0\}}$, we observe that

$$q_f(z_i, z_j)q_f(z_i, z_k) = 1 \Leftrightarrow \begin{cases} y_i = -1, \ y_j = y_k = +1 \text{ and } f(x_i) > f(x_j), f(x_k), \text{ or} \\ y_i = +1, \ y_j = y_k = -1 \text{ and } f(x_i) < f(x_j), f(x_k). \end{cases} \qquad (15)$$

Let us define $\mathcal{E}^+(i)$ and $\mathcal{E}^-(i)$ as

$$\mathcal{E}^+(i) := \{j : y_j = -1, \ f(x_j) > f(x_i)\}, \text{ and } \mathcal{E}^-(i) := \{j : y_j = +1, \ f(x_j) < f(x_i)\}.$$

and their sizes $\kappa_i^+ := |\mathcal{E}^+(i)|$, and $\kappa_i^- := |\mathcal{E}^-(i)|$.

For $i$ such that $y_i = 1$, $\kappa_i^+$ is the number of negative instances that have been scored higher than $x_i$ by $f$. From (15), we see that the contribution of $i$ to the last term of (14) corresponds to the number $\kappa_i^+(\kappa_i^+ - 1)$ of ordered pairs of indices in $\mathcal{E}^+(i)$ (similarly for $\kappa_i^-$, with $y_i = -1$). Henceforth:

$$\sum_{i \neq j \neq k} q_f(z_i, z_j)q_f(z_i, z_k) = \sum_{i: y_i = +1} \kappa_i^+(\kappa_i^+ - 1) + \sum_{i: y_i = -1} \kappa_i^-(\kappa_i^- - 1).$$

A simple way to compute the first sum (on $i$ such that $y_i = +1$) is to sort and visit the data by descending order of scores and then to incrementally compute the $\kappa_i^+$'s and the corresponding sum: when a negative instance is encountered, $\kappa_i^+$ is incremented by 1 and when a positive instance is visited, $\kappa_i^+(\kappa_i^+ - 1)$ is added to the current sum. An identical reasoning works for the second sum.

The cost of computing $\hat{\Sigma}_{q_f}$ is therefore that of sorting the scores, which has cost $O(n \ln n)$. $\qquad \square$

## 4 Applications and Discussion

Here, we mention potential applications of the new empirical inequalities we have just presented.

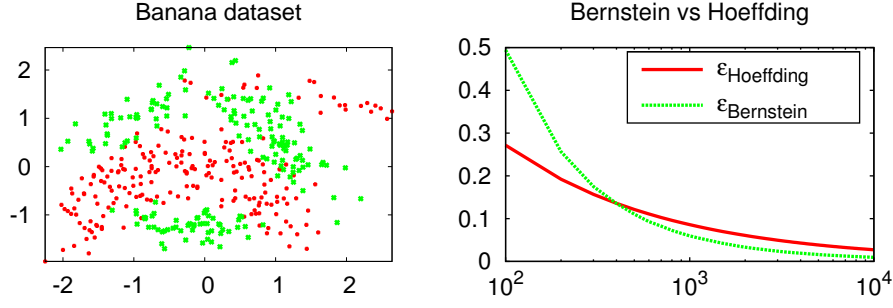

Figure 2: Left: UCI banana dataset, data labelled $+1$ $(-1)$ in red (green). Right: half the confidence interval of the Hoeffding bound and that of the empirical Bernstein bound as functions of $n_{\text{test}}$.

## 4.1 Test Set Bounds

A direct use of the empirical Bernstein inequalities is to draw test set bounds. In this scenario, a sample $\underline{Z}_n$ is split into a training set $\underline{Z}_{\text{train}} := \underline{Z}_{1:n_{\text{train}}}$ of $n_{\text{train}}$ data and a hold-out set $\underline{Z}_{\text{test}} := \underline{Z}_{n_{\text{train}}+1:n}$ of size $n_{\text{test}}$. $\underline{Z}_{\text{train}}$ is used to train a model $f$ that minimizes an empirical risk based on a U-statistic inducing loss (such as in (1) or (2)) and $\underline{Z}_{\text{test}}$ is used to compute a confidence interval on the expected risk of $f$. For instance, if we consider the bipartite ranking problem, the loss is $\ell^{\text{rank}}$, the corresponding kernel is $q_f(Z, Z') = \ell^{\text{rank}}(f, Z, Z')$, and, with probability at least $1 - \delta$

$$R_{\ell^{\text{rank}}}(f) \leq \hat{R}_{\ell^{\text{rank}}}(f, \underline{Z}_{\text{test}}) + \sqrt{\frac{4\hat{\Sigma}^2_{q_f}(\underline{Z}_{\text{test}}) \ln(4/\delta)}{n_{\text{test}}}} + \frac{10}{n_{\text{test}}} \ln\frac{4}{\delta}, \tag{16}$$

where $\hat{\Sigma}^2_{q_f}(\underline{Z}_{\text{test}})$ is naturally the empirical variance of $q_f$ computed on $\underline{Z}_{\text{test}}$.

Figure 2 displays the behavior of such test set bounds as $n_{\text{test}}$ grows for the UCI banana dataset. To produce this plot, we have learned a linear scoring function $f(\cdot) = \langle \mathbf{w}, \cdot \rangle$ by minimizing

$$\lambda\|\mathbf{w}\|^2 + \sum_{i \neq j} \left(1 - (Y_i - Y_j)\langle \mathbf{w}, X_i - X_j\rangle\right)^2$$

for $\lambda = 1.0$. Of course, a purely linear scoring function would not make it possible to achieve good ranking accuracy so we in fact work in the reproducing kernel hilbert space associated with the Gaussian kernel $k(x, x') = \exp(-\|x - x'\|^2/2)$. We train our scoring function on $n_{\text{train}} = 1000$ data points and evaluate the test set bound on $n_{\text{test}} = 100, 500, 1000, 5000, 10000$ data points. Figure 2 (right) reports the size of half the confidence interval of the Hoeffding bound (5) and that of the empirical Bernstein bound given in (16). Just as in the situation described in Example 1, the use of variance information gives rise to smaller confidence intervals, even for moderate sizes of test sets.

## 4.2 Online Ranking and Empirical Racing Algorithms

Another application that we would like to describe is online bipartite ranking. Due to space limitation, we only provide the main ideas on how we think our empirical tail inequalities and the efficient computation of the variance estimates we propose might be particularly useful in this scenario.

First, let us precise what we mean by online bipartite ranking. Obviously, this means that $\mathcal{Y} = \{-1, +1\}$ and that the loss of interest is $\ell^{\text{rank}}$. In addition, it means that given a training set $\underline{Z} = \{Z_i := (X_i, Y_i)\}_{i=1}^n$ the learning procedure will process the data of $\underline{Z}$ incrementally to give rise to hypotheses $f_1, f_2, \ldots, f_T$. As $\ell^{\text{rank}}$ entails a kernel of order $m = 2$, we assume that $n = 2T$ and that we process the data from $\underline{Z}$ pairs by pairs, i.e. $(Z_1, Z_2)$ are used to learn $f_1$, $(Z_3, Z_4)$ and $f_1$ are used to learn $f_2$ and, more generally, $(Z_{2t-1}, Z_{2t})$ and $f_{t-1}$ are used to produce $f_t$ (there exist more clever ways to handle the data but this goes out of the scope of the present paper). We do not specify any learning algorithm but we may imagine trying to minimize a penalized empirical risk based on the surrogate loss $\ell^{\text{sur}}$: if linear functions $f(\cdot) = \langle \mathbf{w}, \cdot \rangle$ are considered and a penalization

like $\|\mathbf{w}\|^2$ is used then the optimization problem to solve is of the same form as in the batch case:

$$\lambda\|\mathbf{w}\|^2 + \sum_{i \neq j}\left(1 - (Y_i - Y_j)\langle \mathbf{w}, X_i - X_j\rangle\right)^2,$$

but is solved incrementally here. Rank-1 update formulas for inverses of matrices easily provide means to incrementally solve this problem as new data arrive (this is the main reason why we have mentioned this surrogate function).

As evoked by [5], a nice feature of online learning is that the expected risk of hypothesis $f_t$ can be estimated on the $n - 2t$ examples of $\underline{Z}$ it was not trained on. Namely, when $2\tau$ data have been processed, there exist $\tau$ hypotheses $f_1, \ldots, f_\tau$ and, for $t < \tau$, with probability at least $1 - \delta$:

$$\left| R_{\ell^{\text{rank}}}(f_t) - \hat{R}_{\ell^{\text{rank}}}(f_t, \underline{Z}_{2t:2\tau})) \right| \leq \sqrt{\frac{2\hat{\Sigma}_{q_f}^2(\underline{Z}_{2t:2\tau})\ln(4/\delta)}{\tau - t}} + \frac{5}{\tau - t}\ln\frac{4}{\delta}.$$

If one wants to have these confidence intervals to *simultaneously* hold for all $t$ and all $\tau$ with probability $1 - \delta$, basic computations to calculate the number of pairs $(t, \tau)$, with $1 \leq t < \tau \leq n$ show that it suffices to adjust $\delta$ to $4\delta/(n+1)^2$. Hence, with probability at least $1 - \delta$: $\forall 1 \leq t < \tau \leq n$,

$$\left| R_{\ell^{\text{rank}}}(f_t) - \hat{R}_{\ell^{\text{rank}}}(f_t, \underline{Z}_{2t:2\tau})) \right| \leq \sqrt{\frac{4\hat{\Sigma}_{q_f}^2(\underline{Z}_{2t:2\tau})\ln((n+1)/\delta)}{\tau - t}} + \frac{10}{\tau - t}\ln\frac{n+1}{\delta}. \qquad (17)$$

We would like to draw the attention of the reader on two features: one has to do with statistical considerations and the other with algorithmic ones. First, if the confidence intervals simultaneously hold for all $t$ and all $\tau$ as in (17), it is possible, as the online learning process goes through, to discard the hypotheses $f_t$ which have their lower bound (according to (17)) on $R_{\ell^{\text{rank}}}(f_t)$ that is higher than the upper bound (according to (17) as well) on $R_{\ell^{\text{rank}}}(f_{t'})$ for some other hypothesis $f_{t'}$. This corresponds to a *racing algorithm* as described in [12]. Theoretically analyzing the relevance of such a race can be easily done with the results of [14], which deal with empirical Bernstein racing, but for non-U-statistics. This full analysis will be provided in a long version of the present paper. Second, it is algorithmically possible to preserve some efficiency in computing the various variance estimates through the online learning process: these computations rely on sorting arguments, and it is possible to take advantage of structures like binary search trees such as AVL trees, that are precisely designed to efficiently maintain and update sorted lists of numbers. The remaining question is whether it is possible to have *shared* such structures to summarize the sorted lists of scores for various hypotheses (recall that the scores are computed on the same data). This will be the subject of further research.

## 5 Conclusion

We have proposed new empirical Bernstein inequalities designed for U-statistics. They generalize the empirical inequalities of [13] and [3] while they merely result from two applications of the same non-empirical tail inequality for U-statistics. We also show how, in the bipartite ranking situation, the empirical variance can be efficiently computed. We mention potential applications, with illustrative results for the case of test set bounds in the realm of bipartite ranking. In addition to the possible extensions discussed in the previous section, we wonder whether it is possible to draw similar empirical inequalities for other types of rich statistics such as, e.g., linear rank statistics [8]. Obviously, we plan to work on establishing generalization bounds derived from the new concentration inequalities presented. This would require to carefully define a sound notion of capacity for U-statistic-based classes of functions (inspired, for example, from localized Rademacher complexities). Such new bounds would be compared with those proposed in [1, 6, 7, 15] for the bipartite ranking and/or pairwise classification problems. Finally, we also plan to carry out intensive simulations —in particular for the task of online ranking— to get even more insights on the relevance of our contribution.

**Acknowledgments**

This work is partially supported by the IST Program of the EC, under the FP7 Pascal 2 Network of Excellence, ICT-216886-NOE. LR is partially supported by the ANR project ASAP.

## References

[1] S. Agarwal, T. Graepel, R. Herbrich, S. Har-Peled, and D. Roth. Generalization Bounds for the Area under the ROC Curve. *Journal of Machine Learning Research*, 6:393–425, 2005.

[2] M. A. Arcones. A bernstein-type inequality for u-statistics and u-processes. *Statistics & probability letters*, 22(3):239–247, 1995.

[3] J.-Y. Audibert, R. Munos, and C. Szepesvári. Tuning bandit algorithms in stochastic environments. In *ALT '07: Proceedings of the 18th international conference on Algorithmic Learning Theory*, pages 150–165, Berlin, Heidelberg, 2007. Springer-Verlag.

[4] S. Boucheron, O. Bousquet, and G. Lugosi. Theory of classification : A survey of some recent advances. *ESAIM. P&S*, 9:323–375, 2005.

[5] N. Cesa-Bianchi, A. Conconi, and C. Gentile. On the generalization ability of online learning algorithms. *IEEE Transactions on Information Theory*, 50(9):2050–2057, 2004.

[6] S. Clémençon, G. Lugosi, and N. Vayatis. Ranking and empirical minimization of u -statistics. *The Annals of Statistics*, 36(2):844–874, April 2008.

[7] Y. Freund, R. Iyer, R.E. Schapire, and Y. Singer. An efficient boosting algorithm for combining preferences. *Journal of Machine Learning Research*, 4:933–969, 2003.

[8] J. Hájek and Z. Sidák. *Theory of Rank Tests*. Academic Press, 1967.

[9] J. A. Hanley and B. J. Mcneil. The meaning and use of the area under a receiver operating characteristic (roc) curve. *Radiology*, 143(1):29–36, April 1982.

[10] W. Hoeffding. A Class of Statistics with Asymptotically Normal Distribution. *Annals of Mathematical Statistics*, 19(3):293–325, 1948.

[11] W. Hoeffding. Probability inequalities for sums of bounded random variables. *Journal of the American Statistical Association*, 58(301):13–30, 1963.

[12] O. Maron and A. Moore. Hoeffding races: Accelerating model selection search for classification and function approximation. In *Adv. in Neural Information Processing Systems NIPS 93*, pages 59–66, 1993.

[13] A. Maurer and M. Pontil. Empirical bernstein bounds and sample-variance penalization. In *COLT 09: Proc. of The 22nd Annual Conference on Learning Theory*, 2009.

[14] V. Mnih, C. Szepesvári, and J.-Y. Audibert. Empirical bernstein stopping. In *ICML '08: Proceedings of the 25th international conference on Machine learning*, pages 672–679, New York, NY, USA, 2008. ACM.

[15] C. Rudin and R. E. Schapire. Margin-based ranking and an equivalence between AdaBoost and RankBoost. *Journal of Machine Learning Research*, 10:2193–2232, Oct 2009.

[16] R. J. Serfling. *Approximation theorems of mathematical statistics*. J. Wiley & Sons, 1980.

